# On Lifting the Gibbs Sampling Algorithm

**Deepak Venugopal**
Department of Computer Science
The University of Texas at Dallas
Richardson, TX, 75080, USA
dxv021000@utdallas.edu

**Vibhav Gogate**
Department of Computer Science
The University of Texas at Dallas
Richardson, TX, 75080, USA
vgogate@hlt.utdallas.edu

## Abstract

First-order probabilistic models combine the power of first-order logic, the de facto tool for handling relational structure, with probabilistic graphical models, the de facto tool for handling uncertainty. Lifted probabilistic inference algorithms for them have been the subject of much recent research. The main idea in these algorithms is to improve the accuracy and scalability of existing graphical models' inference algorithms by exploiting symmetry in the first-order representation. In this paper, we consider blocked Gibbs sampling, an advanced MCMC scheme, and lift it to the first-order level. We propose to achieve this by partitioning the first-order atoms in the model into a set of disjoint clusters such that exact lifted inference is polynomial in each cluster given an assignment to all other atoms not in the cluster. We propose an approach for constructing the clusters and show how it can be used to trade accuracy with computational complexity in a principled manner. Our experimental evaluation shows that lifted Gibbs sampling is superior to the propositional algorithm in terms of accuracy, scalability and convergence.

## 1  Introduction

Modeling large, complex, real-world domains requires the ability to handle both rich relational structure and large amount of uncertainty. Unfortunately, the two existing representation and reasoning tools of choice – probabilistic graphical models (PGMs) and first-order logic – are unable to effectively handle both. PGMs can compactly represent and reason about uncertainty. However, they are propositional and thus ill-equipped to handle relational structure. First-order logic can effectively handle relational structure. However, it has no representation for uncertainty. Therefore, combining the representation and reasoning power of first-order logic with PGMs is a worthwhile goal. Statistical relational learning (SRL) [7] is an emerging field which attempts to do just that.

The key task in SRL is inference - the problem of answering a query given an SRL model. In principle, we can simply ground (propositionalize) the given SRL model to yield a PGM and thereby solve the inference problem in SRL by reducing it to inference over PGMs. This approach is problematic and impractical, however, because the PGMs obtained by grounding a SRL model can be substantially large, having millions of variables and billions of features; existing inference algorithms for PGMs are unable to handle problems at this scale. An alternative approach, which has gained prominence since the work of Poole [25] is lifted or first-order inference. The main idea, which is similar to theorem proving in first-order logic, is to take a propositional inference algorithm and exploit symmetry in its execution by performing inference over a group of identical or interchangeable random variables. The algorithms are called lifted algorithms because they identify symmetry by consulting the first-order representation without grounding the model.

Several lifted algorithms have been proposed to date. Prominent exact algorithms are first-order variable elimination [25] and its extensions [2, 23], which lift the variable elimination algorithm, and probabilistic theorem proving (PTP) [8] which lifts the weighted model counting algorithm [1, 29]. Notable approximate inference algorithms are lifted Belief propagation [30] and lifted importance sampling [8, 9], which lift belief propagation [20] and importance sampling respectively.

In this paper, we lift blocked Gibbs sampling, an advanced MCMC technique. Blocked Gibbs sampling improves upon the Gibbs sampling algorithm by grouping variables (each group is called a block) and then jointly sampling all variables in the block [10, 16]. Blocking improves the mixing time and as a result improves both the accuracy and convergence of Gibbs sampling. The difficulty is that to jointly sample variables in a block, we need to compute a joint distribution over them. This is typically exponential in the treewidth of the ground network projected on the block.

Several earlier papers have attempted to exploit relational or first-order structure in MCMC sampling. Notable examples are lazy MC-SAT [27], Metropolis-Hastings MCMC for Bayesian logic (BLOG) [18], typed MCMC [14] and orbital MCMC [21]. Unfortunately, none of the aforementioned techniques are truly lifted. In particular, they do not exploit first-order structure to the fullest extent. In fact, lifting a generic MCMC technique is difficult because at each point in order to ensure convergence to the desired stationary distribution one has to maintain an assignment to all random variables in the ground network. We circumvent these issues by lifting the blocked Gibbs sampling algorithm, which as we show is more amenable to lifting.

Our main idea in applying the blocking approach to SRL models is to partition the set of first-order atoms in the model into disjoint clusters such that PTP (an exact lifted inference scheme) is feasible in each cluster given an assignment to all other atoms not in the cluster. Given such a set of clusters, we show that Gibbs sampling is essentially a message passing algorithm over the cluster graph formed by connecting clusters that have atoms that are in the Markov blanket of each other. Each message from a sender to a receiving cluster is a truth assignment to all ground atoms that are in the Markov blanket of the receiving cluster. We show how to store this message compactly by taking advantage of the first-order representation yielding a lifted MCMC algorithm.

We present experimental results comparing the performance of lifted blocked Gibbs sampling with (propositional) blocked Gibbs sampling, MC-SAT [26, 27] and Lifted BP [30] on various benchmark SRL models. Our experiments show that lifted Gibbs sampling is superior to blocked Gibbs sampling and MC-SAT in terms of convergence, accuracy and scalability. It is also more accurate than lifted BP on some instances.

## 2 Notation and Preliminaries

In this section, we describe notation and preliminaries on propositional logic, first-order logic, Markov logic networks and Gibbs sampling. For more details, refer to [3, 13, 15].

The language of propositional logic consists of atomic sentences called propositions or atoms, and logical connectives such as $\wedge$ (conjunction), $\vee$ (disjunction), $\neg$ (negation), $\Rightarrow$ (implication) and $\Leftrightarrow$ (equivalence). Each proposition takes values from the binary domain $\{\texttt{False}, \texttt{True}\}$ (or $\{0, 1\}$). A propositional formula $f$ is an atom, or any complex formula that can be constructed from atoms using logical connectives. For example, $\texttt{A}$, $\texttt{B}$ and $\texttt{C}$ are propositional atoms and $f = \texttt{A} \vee \neg\texttt{B} \wedge \texttt{C}$ is a propositional formula. A *knowledge base* (KB) is a set of formulas. A *world* is a truth assignment to all atoms in the KB.

First-order logic (FOL) generalizes propositional logic by allowing atoms to have internal structure; an atom in FOL is a predicate that represents relations between objects. A predicate consists of a predicate symbol, denoted by Monospace fonts, e.g., $\texttt{Friends}$, $\texttt{Smokes}$, etc., followed by a parenthesized list of arguments called *terms*. A term is a logical variable, denoted by lower case letters such as $x$, $y$, $z$, etc., or a constant, denoted by upper case letters such as $X$, $Y$, $Z$, etc. We assume that each logical variable, e.g., $x$ is typed and takes values over a finite set $\Delta_x$. The language of FOL also includes two quantifiers: $\forall$ (universal) and $\exists$ (existential) which express properties of an entire collection of objects. A formula in first order logic is a predicate (atom), or any complex sentence that can be constructed from atoms using logical connectives and quantifiers. For example, the formula $\forall x\ \texttt{Smokes}(x) \Rightarrow \texttt{Asthma}(x)$ states that all persons who smoke have asthma. $\exists x\ \texttt{Cancer}(x)$ states that there exists a person $x$ who has cancer. A *first-order KB* is a set of first-order formulas.

In this paper, we use a subset of FOL which has no function symbols, equality constraints or existential quantifiers. We also assume that domains are finite (and therefore function-free) and that there is a one-to-one mapping between constants and objects in the domain (Herbrand interpretations). We assume that each formula $f$ is of the form $\forall \mathbf{x}\ f$, where $\mathbf{x}$ are the set of variables in $f$ and $f$ is a conjunction or disjunction of literals; each literal being an atom or its negation. For brevity, we will drop $\forall$ from all the formulas. Given variables $\mathbf{x} = \{x_1, \ldots, x_n\}$ and constants $\mathbf{X} = \{X_1, \ldots, X_n\}$

where $X_i \in \Delta_{x_i}$, $f[\mathbf{X}/\mathbf{x}]$ is obtained by substituting every occurrence of variable $x_i$ in $f$ with $X_i$. A ground formula is a formula obtained by substituting all of its variables with a constant. A ground KB is a KB containing all possible groundings of all of its formulas. For example, the grounding of a KB containing one formula, $\texttt{Smokes}(x) \Rightarrow \texttt{Asthma}(x)$ where $\Delta_x = \{Ana, Bob\}$, is a KB containing two formulas: $\texttt{Smokes}(Ana) \Rightarrow \texttt{Asthma}(Ana)$ and $\texttt{Smokes}(Bob) \Rightarrow \texttt{Asthma}(Bob)$. A *world* in FOL is a truth assignment to all atoms in its grounding.

Markov logic [3] extends FOL by softening the hard constraints expressed by the formulas and is arguably the most popular modeling language for SRL. A soft formula or a weighted formula is a pair $(f, w)$ where $f$ is a formula in FOL and $w$ is a real-number. A Markov logic network (MLN), denoted by $\mathcal{M}$, is a set of weighted formulas $(f_i, w_i)$. Given a set of constants that represent objects in the domain, a Markov logic network defines a Markov network or a log-linear model. The Markov network is obtained by grounding the weighted first-order knowledge base and represents the following probability distribution.

$$P_{\mathcal{M}}(\omega) = \frac{1}{Z(\mathcal{M})} \exp\left(\sum_i w_i N(f_i, \omega)\right) \tag{1}$$

where $\omega$ is a world, $N(f_i, \omega)$ is the number of groundings of $f_i$ that evaluate to $\texttt{True}$ in the world $\omega$ and $Z(\mathcal{M})$ is a normalization constant or the partition function.

In this paper, we assume that the input MLN to our algorithm is in normal form [11, 19]. We require this for simplicity of exposition. Our main algorithm can be easily modified to work with other canonical forms such as parfactors [25] and first order CNFs with substitution constraints [8]. However, its specification becomes much more complicated and messy. A *normal* MLN [11] is an MLN that satisfies the following two properties: (1) There are no constants in any formula, and (2) If two distinct atoms with the same predicate symbol have variables $x$ and $y$ in the same position then $\Delta_x = \Delta_y$. Note that in a normal MLN, we assume that the terms in each atom are ordered and therefore we can identify each term by its position in the order.

## 2.1 Gibbs Sampling and Blocking

Given an MLN, a set of query atoms and evidence, we can adapt the basic (propositional) Gibbs sampling [6] algorithm for computing the marginal probabilities of query atoms given evidence as follows. First, we ground all the formulas in the MLN, yielding a Markov network. Second, we instantiate all the evidence atoms in the network. Assume that the resulting evidence-instantiated network is defined over a set of variables $\mathbf{X}$. Third, we generate $N$ samples $(\bar{\mathbf{x}}^{(1)}, \ldots, \bar{\mathbf{x}}^{(N)})$ (a sample is a truth assignment to all random variables in the Markov network) as follows. We begin with a random assignment to all variables, yielding $\bar{\mathbf{x}}^{(0)}$. Then for $t = 1, \ldots, N$, we perform the following steps. Let $(X_1, \ldots, X_n)$ be an arbitrary ordering of variables in $\mathbf{X}$. Then, for $i = 1$ to $n$, we generate a new value $\bar{x}_i^{(t)}$ for $X_i$ by sampling a value from the distribution $P(X_i | \bar{x}_1^t, \ldots, \bar{x}_{i-1}^t, \bar{x}_{i+1}^{(t-1)}, \ldots, \bar{x}_n^{(t-1)})$. (This is often called systematic scan Gibbs sampling. An alternative approach is random scan Gibbs sampling which often converges faster than systematic scan Gibbs sampling). For conciseness, we will write $P(X_i | \bar{\mathbf{x}}_{-i}^{(t)}) = P(X_i | \bar{x}_1^t, \ldots, \bar{x}_{i-1}^t, \bar{x}_{i+1}^{(t-1)}, \ldots, \bar{x}_n^{(t-1)})$. Once the required $N$ samples are generated, we can use them to answer any query over the model. In particular, the marginal probability for each variable can be estimated by averaging the conditional marginals:

$$\widehat{P}(\bar{x}_i) = \frac{1}{N} \sum_{t=1}^{N} P(\bar{x}_i | \bar{\mathbf{x}}_{-i}^{(t)})$$

Note that in Markov networks, $P(X_i | \bar{\mathbf{x}}_{-i}^{(t)}) = P(X_i | \bar{\mathbf{x}}_{-i, MB(X_i)}^{(t)})$ where $MB(X_i)$ is the Markov Blanket (the set of variables that share a function with $X_i$) of $X_i$ and $\bar{\mathbf{x}}_{-i, MB(X_i)}^{(t)}$ is the projection of $\bar{\mathbf{x}}_{-i}^{(t)}$ on $MB(X_i)$.

The sampling distribution of Gibbs sampling converges to the posterior distribution (the distribution associated with the evidence instantiated Markov network) as the number of samples increases because the resulting Markov chain is guaranteed to be aperiodic and ergodic (see [15] for details).

The main idea in blocked Gibbs sampling [10] is grouping variables to form a block, and then jointly sampling all variables in a block given an assignment to all other variables not in the block.

Blocking improves mixing yielding a more accurate sampling algorithm [15]. However, the computational complexity of jointly sampling all variables in a block typically increases with the treewidth of the Markov network projected on the block. Thus, in practice, given time and memory resource constraints, the main issue in blocked Gibbs sampling is finding the right balance between computational complexity and accuracy.

## 3 Our Approach

We illustrate the key ideas in our approach using an example MLN having two weighted formulas: $R(x, y) \vee S(y, z), w_1$ and $S(y, z) \vee T(z, u), w_2$. Note that the problem of computing the partition function of this MLN for arbitrary domain sizes is non-trivial; it cannot be polynomially solved using existing exact lifted approaches such as PTP [8] and lifted VE [2].

Our main idea is to partition the set of atoms into disjoint blocks (clusters) such that PTP is polynomial in each cluster and then sample all atoms in the cluster jointly. PTP is polynomial if we can recursively apply its two lifting rules (defined next), the *power rule* and the *generalized binomial rule*, until the treewidth of the remaining ground network is bounded by a constant.

The power rule is based on the concept of a decomposer. Given a normal MLN $\mathcal{M}$, a set of logical variables, denoted by $\mathbf{x}$, is called a *decomposer* if it satisfies the following two conditions: (i) Every atom in $\mathcal{M}$ contains exactly one variable from $\mathbf{x}$, and (ii) For any predicate symbol $R$, there exists a position s.t. variables from $\mathbf{x}$ only appear at that position in atoms of $R$. Given a decomposer $\mathbf{x}$, it is easy to show that $Z(\mathcal{M}) = [Z(\mathcal{M}[X/\mathbf{x}])]^{|\Delta_x|}$ where $x \in \mathbf{x}$ and $\mathcal{M}[X/\mathbf{x}]$ is the MLN obtained by substituting all logical variables $\mathbf{x}$ in $\mathcal{M}$ by the same constant $X \in \Delta_x$ and then converting the resulting MLN to a normal MLN. Note that for any two variables $x, y$ in $\mathbf{x}$, $\Delta_x = \Delta_y$ by normality.

The generalized binomial rule is used to sample singleton atoms efficiently (the rule also requires that the atom is not involved in self-joins, i.e., it does not appear more than once in the same formula). Given a normal MLN $\mathcal{M}$ having a singleton atom $R(x)$, we can show that $Z(\mathcal{M}) = \sum_{i=0}^{|\Delta_x|} \binom{|\Delta_x|}{i} Z(\mathcal{M}|\bar{R}^i) w(i) 2^{p(i)}$ where $\bar{R}^i$ is a sample of $R$ s.t. exactly $i$ tuples are set to True. $\mathcal{M}|\bar{R}^i$ is the MLN obtained from $\mathcal{M}$ by performing the following steps in order: (i) Ground all $R(x)$ and set its groundings to have the same assignment as $R^i$, (ii) Delete formulas that evaluate to either True or False, (iii) Delete all groundings of $R(x)$ and (iv) Convert the resulting MLN to a normal MLN. $w(i)$ is the exponentiated sum of the weights of formulas that evaluate to True and $p(i)$ is the number of ground atoms that are removed from the MLN as a result of removing formulas (these are essentially don't care atoms which can be assigned to either True or False).

Now, let us apply the clustering idea to our example MLN. Let us put each first-order atom in a cluster by itself, namely we have three clusters: $R(x, y)$, $S(y, z)$ and $T(z, u)$ (see Figure 1(a)). Note that each (first-order) cluster represents all groundings of all atoms in the cluster. To perform Gibbs sampling over this clustering, we need to compute three conditional distributions: $P(R(x, y)|\bar{S}(y, z), \bar{T}(z, u))$, $P(S(y, z)|\bar{R}(x, y), \bar{T}(z, u))$ and $P(T(z, u)|\bar{R}(x, y), \bar{S}(y, z))$ where $\bar{R}(x, y)$ denotes a truth assignment to all possible groundings of $R$. Let the domain size of each variable be $n$. Naively, given an assignment to all other atoms not in the cluster, we will need $O(2^{n^2})$ time and space for computing and specifying

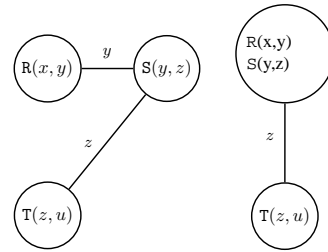

(a) Clustering 1      (b) Clustering 2

Figure 1: Two possible clusterings for lifted blocked Gibbs sampling on the example MLN having two weighted formulas.

the joint distribution at each cluster. This is because there are $n^2$ ground atoms associated with each cluster. Notice however that all groundings of each first-order atom are conditionally independent of each other given a truth assignment to all other atoms. In other words, we can apply PTP here and compute each conditional distribution in $O(n^3)$ time and space (since there are $n^3$ groundings of each formula and we need to process each ground formula at least once). Thus, the complexity of sampling all atoms in all clusters is $O(n^3)$. Note that the complexity of sampling all variables using propositional Gibbs sampling is also $O(n^3)$.

Now, let us consider an alternative clustering in which we have two clusters as shown in Figure 1(b). Intuitively, this clustering is likely to yield better accuracy than the previous one because more

atoms will be sampled jointly. Counter-intuitively, however, as we show next, Clustering 2 will yield a blocked sampler having smaller complexity than the one based on Clustering 1.

To perform blocked Gibbs sampling over Clustering 2, we need to compute two distributions $P(\mathtt{R}(x,y),\mathtt{S}(y,z)|\bar{\mathtt{T}}(z,u))$, $P(\mathtt{T}(z,u)|\bar{\mathtt{R}}(x,y),\bar{\mathtt{S}}(y,z))$. Let us see how PTP will compute $P(\mathtt{R}(x,y),\mathtt{S}(y,z)|\bar{\mathtt{T}}(z,u))$. If we instantiate all groundings of T, we get the following reduced MLN $\{\mathtt{R}(x,y) \vee \mathtt{S}(y,Z_i), w_1\}_{i=1}^n$ and $\{\mathtt{S}(y,Z_i), k_i w_2\}_{i=1}^n$ where $Z_i \in \Delta_z$ and $k_i$ is the number of False groundings of $\mathtt{T}(y,Z_i)$. This MLN contains a decomposer $y$. PTP will now apply the power rule, yielding formulas of the form $\{\mathtt{R}(x,Y) \vee \mathtt{S}(Y,Z_i), w_1\}_{i=1}^n$ and $\{\mathtt{S}(Y,Z_i), k_i w_2\}_{i=1}^n$ where $Y \in \Delta_y$. $\mathtt{R}(x,Y)$ is a singleton atom and therefore applying the generalized binomial rule, we will get $n + 1$ reduced MLNs, each containing $n$ atoms of the form $\{\mathtt{S}(Y,Z_i)\}_{i=1}^n$. These atoms are conditionally independent of each other and a distribution over them can be computed in $O(n)$ time. Thus, the complexity of computing $P(\mathtt{R}(x,y),\mathtt{S}(y,z)|\bar{\mathtt{T}}(z,u))$ is $O(n^2)$. Samples for R and S can be generated from $P(\mathtt{R}(x,y),\mathtt{S}(y,z)|\bar{\mathtt{T}}(z,u))$ in $O(n^2)$ time as well. Notice that $P(\mathtt{T}(z,u)|\bar{\mathtt{R}}(x,y),\bar{\mathtt{S}}(y,z)) = P(\mathtt{T}(z,u)|\bar{\mathtt{S}}(y,z))$ because R is not in the Markov blanket of T. This distribution can also be computed in $O(n^2)$ time. Therefore, the complexity of sampling all atoms using the clustering shown in Figure 1(b) is $O(n^2)$.

*Space Complexity:* For Clustering 2, notice that to compute the conditional distribution $P(\mathtt{R}(x,y),\mathtt{S}(y,z)|\bar{\mathtt{T}}(z,u))$, we only need to know how many groundings of $\mathtt{T}(Z_i,u)$ are True in $\bar{\mathtt{T}}(z,u)$ for all $Z_i \in \Delta_z$. Cluster $\mathtt{T}(z,u)$ can share this information with its neighbor using only $O(n)$ space. Similarly, to compute $P(\mathtt{T}(z,u)|\bar{\mathtt{S}}(y,z))$ we only need to know how many groundings of $\mathtt{S}(y,Z_i)$ are True in $\bar{\mathtt{S}}(y,z)$ for all $Z_i \in \Delta_z$. This requires $O(n)$ space and thus the overall space complexity of Clustering 2 is $O(n)$. On the other hand, the space complexity of Gibbs sampling over Clustering 1 is $O(n^2)$.

## 4 The Lifted Blocked Gibbs Sampling Algorithm

Next, we will formalize the discussion in the previous section yielding a lifted blocked Gibbs sampling algorithm. We begin with some required definitions.

We define a *cluster* as a set of first order atoms (these atoms will be sampled jointly in a lifted Gibbs sampling iteration). Given a set of disjoint clusters $\{C_1, \ldots, C_m\}$, the Markov blanket of a cluster $C_i$ is the set of clusters that have at least one atom that is in the Markov blanket of an atom in $C_i$. Given a MLN $\mathcal{M}$, the *Gibbs cluster graph* is a graph $G$ (each vertex of $G$ is a cluster) such that: (i) Each atom in the MLN is in exactly one cluster of $G$ (ii) Two clusters $C_i$ and $C_j$ in $G$ are connected by an edge if $C_j$ is in the Markov blanket of $C_i$. Note that by definition if $C_i$ is in the Markov blanket of $C_j$, then $C_j$ is in the Markov blanket of $C_i$.

---

**Algorithm 1**: Lifted Blocked Gibbs Sampling

**Input**: A normal MLN $\mathcal{M}$, a Gibbs cluster graph $G$, an integer $N$ and a set of query atoms

**Output**: A Marginal Distribution over the query atoms

1 **begin**
2    **for** $t = 1$ *to* $N$ **do**
3      Let $(C_1, \ldots, C_m)$ be an arbitrary ordering of clusters of $G$
     // Gibbs iteration
4      **for** $i = 1$ *to* $m$ **do**
5        $\mathcal{M}(C_i)$ = MLN obtained by instantiating the Markov Blanket of $C_i$ based on the incoming messages
6        Compute $P(C_i)$ by running PTP on $\mathcal{M}(C_i)$
7        Sample a truth assignment to all atoms in $C_i$ from $P(C_i)$
8        Update the estimate of all query atoms in $C_i$
9        Update all outgoing messages from $C_i$
10 **end**

---

The lifted blocked Gibbs sampling algorithm (see Algorithm 1) can be envisioned as a message passing algorithm over a Gibbs cluster graph $G$. Each edge $(C_i, C_j)$ in $G$ stores two messages in each direction. The message from $C_i$ to $C_j$ contains the current truth assignment to all groundings of all atoms (we will discuss how to represent the truth assignment in a lifted manner shortly) that are in the Markov blanket of one or more atoms in $C_i$. We initialize the messages randomly. Then at each Gibbs iteration, we generate a sample over all atoms by sampling the clusters along an ordering $(C_1, \ldots, C_m)$ (Steps 3-10). At each cluster, we first use PTP to compute a conditional joint distribution over all atoms in the cluster given an assignment to atoms in their Markov blanket. This assignment is derived using the incoming messages. Then, we sample all atoms in the cluster from the joint distribution and update the estimate for query atoms in the cluster as well as all outgoing messages. We can prove that:

**Theorem 1.** The Markov chain induced by Algorithm 1 is ergodic and aperiodic and its stationary distribution is the distribution represented by the input normal MLN.

## 4.1  Lifted Message Representation

We say that a representation of truth assignments to the groundings of an atom is lifted if we only specify the number of true (or false) assignments to its full or partial grounding.

**Example 1.** Consider an atom $\mathtt{R}(x,y)$, where $\Delta_x = \{X_1, X_2\}$ and $\Delta_y = \{Y_1, Y_2\}$. We can represent the truth assignment $(\mathtt{R}(X_1, Y_1) = 1, \mathtt{R}(X_1, Y_2) = 0, \mathtt{R}(X_2, Y_1) = 1, \mathtt{R}(X_2, Y_2) = 0)$ in a lifted manner using either an integer 2 or a vector $([Y_1, 2], [Y_2, 0])$. The first representation says that 2 groundings of $\mathtt{R}(x,y)$ are true while the second representation says that 2 groundings of $\mathtt{R}(x, Y_1)$ and 0 groundings of $\mathtt{R}(x, Y_2)$ are true.

Next, we state sufficient conditions for representing a message in a lifted manner while ensuring correctness, summarized in Theorem 2. We begin with a required definition. Given an atom $\mathtt{R}(x_1, \ldots, x_p)$ and a subset of atoms $\{\mathtt{S}_1, \ldots, \mathtt{S}_k\}$ from its Markov blanket, we say that a term at position $i$ in $\mathtt{R}$ is a *shared term* w.r.t. $\{\mathtt{S}_1, \ldots, \mathtt{S}_k\}$ if there exists a formula $f$ such that in $f$, a logical variable appears at position $i$ in $\mathtt{R}$ and in one or more atoms in $\{\mathtt{S}_1, \ldots, \mathtt{S}_k\}$. For instance, in our running example, $y$ (position 2) is a shared term of $\mathtt{R}$ w.r.t. $\{\mathtt{S}\}$ but $x$ (position 1) is not.

**Theorem 2** (**Sufficient Conditions for a Lifted Message Representation**). Given a Gibbs cluster graph $G$ and an MLN $\mathcal{M}$, let $\mathtt{R}$ be an atom in $C_i$ and let $C_j$ be a neighbor of $C_i$ in $G$. Let $S_{\mathtt{R},C_j}$ be the set of atoms formed by taking an intersection between the Markov blanket of $\mathtt{R}$ and the union of the Markov blanket of atoms in $C_j$. Let $\mathbf{x}$ be the set of shared terms of $\mathtt{R}$ w.r.t. $S_{\mathtt{R},C_j} \cup C_j$ and $\mathbf{y}$ be the set of remaining terms in $\mathtt{R}$. Let the outgoing message from $C_i$ to $C_j$ be represented using a vector of $|\Delta_{\mathbf{x}}|$ pairs of the form $[\mathbf{X}_k, r_k]$ where $\Delta_{\mathbf{x}}$ is the Cartesian product of the domains of all terms in $\mathbf{x}$, $\mathbf{X}_k \in \Delta_{\mathbf{x}}$ is the $k$-th element in $\Delta_{\mathbf{x}}$ and $r_k$ is the number of groundings of $\mathtt{R}(\mathbf{X}_k, \mathbf{y})$ that are true in the current assignment. If all messages in the lifted Blocked Gibbs sampling algorithm (Algorithm 1) use the aforementioned representation, then the stationary distribution of the Markov chain induced by the algorithm is the distribution represented by the input normal MLN.

*Proof.* (Sketch). The generalized Binomial rule states that all MLNs obtained by conditioning on a singleton atom $\mathtt{S}$ with exactly $k$ of its groundings set to true are equivalent to each other. In other words, in order to compute the distribution represented by the MLN conditioned on $\mathtt{S}$, we only need to know how many groundings of $\mathtt{S}$ are set to true. Next, we will show that the atom obtained by (partially) grounding the shared terms $\mathbf{x}$ of an atom $\mathtt{R}$ in cluster $C_i$, namely $\mathtt{R}(\mathbf{X}_k, \mathbf{y})$ (where $\mathbf{y}$ is the set of terms of $\mathtt{R}$ that are not shared) is equivalent to a singleton atom and therefore knowing the number of groundings of $\mathtt{R}(\mathbf{X}_k, \mathbf{y})$ that are set to true is sufficient to compute the joint distribution over the atoms in cluster $C_j$, where $C_i$ and $C_j$ are neighbors in $G$.

Consider the MLN $\mathcal{M}'$ which is obtained from $\mathcal{M}$ by first removing all formulas that do not mention atoms in $C_j$ and then (partially) grounding all the shared terms of $\mathtt{R}$. Let $y'$ be a logical variable such that its domain $\Delta_{y'} = \Delta_{\mathbf{y}}$, where $\Delta_{\mathbf{y}}$ is the Cartesian product of the domains of all variables in $\mathbf{y}$ and let $\mathtt{R}'_k(y') = \mathtt{R}(\mathbf{X}_k, \mathbf{y})$ where $\mathbf{X}_k \in \Delta_{\mathbf{x}}$ is the $k$-th element in $\Delta_{\mathbf{x}}$. Notice that we can replace each atom $\mathtt{R}(\mathbf{X}_k, \mathbf{y})$ in $\mathcal{M}'$ by $\mathtt{R}'_k(y')$ without changing the associated distribution. Moreover, each atom $\mathtt{R}'_k(y')$ is a singleton and therefore it follows from the generalized Binomial rule that in order to compute the distribution associated with $\mathcal{M}'$ conditioned on $\mathtt{R}'_k(y')$, we only need to know how many of its possible groundings are true. Since $C_i$ sends precisely this information to $C_j$ using the message defined in the statement of this theorem, it follows that the lifted Blocked Gibbs sampling algorithm which uses a lifted message representation is equivalent to the algorithm (Algorithm 1) that uses a propositional representation. Since Algorithm 1 converges to the distribution represented by the MLN (Theorem 1), the proof follows. □

## 4.2  Complexity

Theorem 2 provides a method for representing the messages succinctly by taking advantage of the symmetry at inference time. It also generalizes the ideas presented in the previous section (last paragraph) and helps us bound the space complexity of each message. Formally,

**Theorem 3** (**Space Complexity of a Message**). Given a Gibbs cluster graph $G$ and an MLN $\mathcal{M}$, let the outgoing message from cluster $C_i$ to cluster $C_j$ in $G$ be defined over the set $\{\mathtt{R}_1, \ldots, \mathtt{R}_k\}$ of atoms. Let $\mathbf{x}_i$ denote the set of shared terms of $\mathtt{R}_i$ that satisfy the conditions outlined in Theorem 2. Then, the space complexity of representing the message is $O(\sum_{i=1}^{k} |\Delta_{\mathbf{x}_i}|)$.

Note that the time/space requirements of the algorithm is the sum of the time/space required to run PTP for a cluster and the time/space for the message from the cluster. We can compute the time

and space complexity of PTP at a cluster by running it schematically as follows. We apply the power rule as before but explore only one randomly selected branch in the search tree induced by the generalized binomial rule. Recall that applying the generalized binomial rule will result in $n+1$ recursive calls (i.e, the search tree node has branching factor of $n+1$) where $n$ is the domain size of the singleton atom. If neither the power rule nor the generalized binomial rule can be applied at any point during search, the complexity of PTP is exponential in the treewidth of the remaining ground network. More precisely, the complexity of PTP is $O(\exp(g) \times \exp(w+1))$ where $g$ is the number of times the generalized binomial rule is applied and $w$ is the treewidth (computed heuristically) of the remaining ground network.

## 4.3 Constructing the Gibbs Cluster Graph

---

**Algorithm 2**: Construct Gibbs Cluster Graph

**Input**: A normal MLN $\mathcal{M}$, complexity bounds $\alpha$ and $\beta$
**Output**: A Gibbs cluster graph $G$

1 **begin**
2    **Initialization:** Construct a Gibbs cluster graph $G$ with exactly one atom in each cluster
3    **while** True **do**
4      $F = \emptyset$ // F: Set of feasible cluster graphs
5      **for** all pairs of clusters $C_i$ and $C_j$ in G **do**
6        Merge $C_i$ and $C_j$ yielding a cluster graph $G'$
7        **if** $T(G') \leq T(G)$ and $S(G') \leq S(G)$ **then**
8          Add $G'$ to $F$
9        **else if** $T(G') \leq \alpha$ and $S(G') \leq \beta$ **then**
10          Add $G'$ to $F$
11     If $F$ is empty **return** $G$
12     $G$ = Cluster graph in $F$ that has the maximum $\sum_i \zeta(C_i)$
13 **end**

---

Next, we present a heuristic algorithm for constructing the Gibbs cluster graph. From a computational view point, we want its time and space requirements to be as small as possible. From an approximation quality viewpoint, to improve mixing, we want to jointly sample, i.e., cluster together highly coupled/correlated variables. Formally, we want to

$$\text{Maximize: } \sum_i \zeta(C_i),$$
$$\text{Subject to: } S(G) \leq \alpha, T(G) \leq \beta$$

where $S(G)$ and $T(G)$ denote the time and space requirements of the Gibbs cluster graph $G$, $\zeta(C_i)$ measures the amount of coupling in the cluster $C_i$ of $G$, and parameters $\alpha$ and $\beta$ are used to bound the time and space complexity respectively. In our implementation, we measure coupling using the number of times two atoms appear together in a formula.

The optimization problem is NP-hard in general and therefore we propose to use the greedy approach given in Algorithm 2 for solving it. The algorithm begins by constructing a Gibbs cluster graph in which each first-order atom is in a cluster by itself. Then, in the while loop, the algorithm tries to iteratively improve the cluster graph. At each iteration, given the current cluster graph $G$, for every possible pair of clusters $(C_i, C_j)$ of $G$, the algorithm creates a new cluster graph $G'$ from $G$ by merging $C_i$ and $C_j$. Among these graphs, the algorithm selects the graph that yields the most coupling and at the same time either has smaller complexity than $G$ or satisfies the input complexity bounds $\alpha$ and $\beta$. It then replaces $G$ with the selected graph and iterates until the graph cannot be improved. Note that increasing the cluster size may decrease the complexity of the cluster graph in some cases and therefore we require steps 6 and 7 which add $G'$ to the feasible set if its complexity is smaller than $G$. Also note that the algorithm is not guaranteed to return a cluster graph that satisfies the input complexity bounds, even if such a cluster graph exists. If the algorithm fails then we may have to use local search or dynamic programming; both are computationally expensive.

## 5 Experiments

In this section, we compare the performance of lifted blocked Gibbs sampling (LBG) with (propositional) blocked Gibbs sampling (BG), lazy MC-SAT [26, 27] and lifted belief propagation (LBP) [30]. We experimented with the following four MLNs: (i) A RST MLN having two formulas, $\mathcal{M}_1$ : [R$(x)$ ∨ S$(x,y)$, $w_1$]; [S$(x,y)$ ∨ T$(y,z)$], (ii) A toy Smoker-Asthma-Cancer MLN having three formulas, $\mathcal{M}_3$ : [Asthma$(x)$ → ¬Smokes$(x)$], [Asthma$(x)$ ∧ Friends$(x,y)$ → ¬Smokes$(y)$], [Smoke$(x)$ → Cancer$(x)$], (iii) The example R, S, T MLN defined in Section 3, $\mathcal{M}_3$ and (iv) WEBKB MLN, $\mathcal{M}_4$ used in [17]. Note that the first two MLNs can be solved in polynomial time using PTP while PTP is exponential on $\mathcal{M}_3$ and $\mathcal{M}_4$. For each MLN, we set 10% randomly selected ground atoms as evidence. We varied the number of objects in the domain from 5 to 200. We used a time-bound of 1000 seconds for all algorithms.

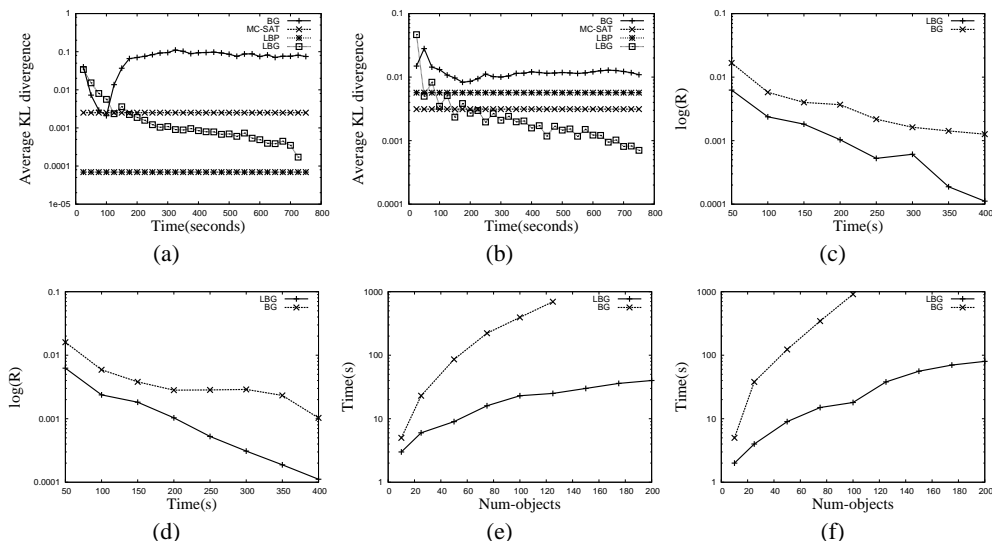

Figure 2: KL divergence as a function of time for: (a) $\mathcal{M}_1$ with 50 objects and (b) $\mathcal{M}_2$ with 50 objects. Convergence diagnostic using Gelman-Rubin statistic ($R$) for (c) $\mathcal{M}_3$ with 25 objects and (d) $\mathcal{M}_4$ with 25 objects. Note that for lifted BP, the values displayed are the ones obtained after the algorithm has converged. Time required by 100 Gibbs iterations as a function of the number of objects for (e) $\mathcal{M}_3$ and (f) $\mathcal{M}_4$.

We implemented LBG and BG in C++ and used alchemy [12] to implement MC-SAT and LBP. For LBG, BG and MC-SAT, we used a burn-in of 100 samples to negate the effects of initialization. For $\mathcal{M}_1$ and $\mathcal{M}_2$, we measure the accuracy using the KL divergence between the estimated marginal probabilities and the true marginal probabilities computed using PTP. Since computing exact marginals of $\mathcal{M}_3$ and $\mathcal{M}_4$ is not feasible, we perform convergence diagnostics for LBG and BG using the Gelman-Rubin statistic [5], denoted by $R$. $R$ measures the disagreement between chains by comparing the between-chain variances with the within-chain variances. The closer the value of $R$ to 1, the better the mixing.

Figure 2 shows the results. Figures 2(a) and 2(b) show the KL divergence as a function of time for $\mathcal{M}_1$ and $\mathcal{M}_2$ respectively. In both cases, LBG converges much faster than BG and MC-SAT and has smaller error. LBP is more accurate than LBG on $\mathcal{M}_1$ while LBG is more accurate than LBP on $\mathcal{M}_2$. Figures 2(c) and 2(d) show $\log(R)$ as a function of time for $\mathcal{M}_3$ and $\mathcal{M}_4$ respectively. We see that the Markov chain associated with LBG mixes much faster than the one associated with BG. To measure scalability, we use running time per Gibbs iteration as a performance metric. Figures 2(e) and 2(f) show the time required by 100 Gibbs iterations as a function of number of objects for $\mathcal{M}_3$ and $\mathcal{M}_4$ respectively. They clearly demonstrates that LBG is more scalable than BG.

# 6   Summary and Future Work

In this paper, we proposed lifted Blocked Gibbs sampling, a new algorithm that improves blocked Gibbs sampling by exploiting relational or first-order structure. Our algorithm operates by constructing a Gibbs cluster graph, which represents a partitioning of atoms into clusters and then performs message passing over the graph. Each message is a truth assignment to the Markov blanket of the cluster and we showed how to represent it in a lifted manner. We proposed an algorithm for constructing the Gibbs cluster graph and showed that it can be used to trade accuracy with computational complexity. Our experiments demonstrate clearly that lifted blocked Gibbs sampling is more accurate and scalable than propositional blocked Gibbs sampling as well as MC-SAT.

Future work includes: lifting Rao-Blackwellised Gibbs sampling; applying our lifting rules to slice sampling [22] and flat histogram MCMC [4]; developing new clustering strategies; etc.

**Acknowledgements:** This research was partly funded by the ARO MURI grant W911NF-08-1-0242. The views and conclusions contained in this document are those of the authors and should not be interpreted as necessarily representing the official policies, either expressed or implied, of ARO or the U.S. Government.

# References

[1] M. Chavira and A. Darwiche. On probabilistic inference by weighted model counting. *Artificial Intelligence*, 172(6-7):772–799, 2008.

[2] R. de Salvo Braz. *Lifted First-Order Probabilistic Inference*. PhD thesis, University of Illinois, Urbana-Champaign, IL, 2007.

[3] P. Domingos and D. Lowd. *Markov Logic: An Interface Layer for Artificial Intelligence*. Morgan & Claypool, San Rafael, CA, 2009.

[4] S. Ermon, C.P. Gomes, A. Sabharwal, and B. Selman. Accelerated Adaptive Markov Chain for Partition Function Computation. In *NIPS*, pages 2744–2752, 2011.

[5] A. Gelman and D. B. Rubin. Inference from iterative simulation using multiple sequences. *Statistical Science*, 7(4):457–472, 1992.

[6] S. Geman and D. Geman. Stochastic relaxation, Gibbs distributions, and the Bayesian restoration of images. *IEEE Transactions on Pattern Analysis and Machine Intelligence*, 6:721–741, 1984.

[7] L. Getoor and B. Taskar, editors. *Introduction to Statistical Relational Learning*. MIT Press, 2007.

[8] V. Gogate and P. Domingos. Probabilistic theorem proving. In *UAI*, pages 256–265, 2011.

[9] V. Gogate, A. Jha, D. Venugopal. Advances in Lifted Importance Sampling. In *AAAI*, pages 1910–1916, 2012.

[10] C. S. Jensen, U. Kjaerulff, and A. Kong. Blocking gibbs sampling in very large probabilistic expert systems. *International Journal of Human Computer Studies. Special Issue on Real-World Applications of Uncertain Reasoning*, 42:647–666, 1993.

[11] A. Jha, V. Gogate, A. Meliou, and D. Suciu. Lifted inference from the other side: The tractable features. In *NIPS*, pages 973–981, 2010.

[12] S. Kok, M. Sumner, M. Richardson, P. Singla, H. Poon, and P. Domingos. The Alchemy system for statistical relational AI. Technical report, Department of Computer Science and Engineering, University of Washington, Seattle, WA, 2006. http://alchemy.cs.washington.edu.

[13] D. Koller and N. Friedman. *Probabilistic Graphical Models: Principles and Techniques*. MIT Press, 2009.

[14] P. Liang, M. I. Jordan, and D. Klein. Type-based MCMC. In *HLT-NAACL*, pages 573–581, 2010.

[15] J. S. Liu. *Monte Carlo Strategies in Scientific Computing*. Springer Publishing Company, Incorporated, 2001.

[16] J. S. Liu, W. H. Wong, and A. Kong. Covariance structure of the Gibbs sampler with applications to the comparison of estimators and augmentation schemes. *Biometrika*, 81:27–40, 1994.

[17] D. Lowd and P. Domingos. Recursive random fields. In *IJCAI*, pages 950–955. 2007.

[18] B. Milch and S. J. Russell. General-purpose MCMC inference over relational structures. In *UAI*, pages 349–358, 2006.

[19] B. Milch, L. S. Zettlemoyer, K. Kersting, M. Haimes, and L. P. Kaelbling. Lifted probabilistic inference with counting formulas. In *AAAI*, pages 1062–1068, 2008.

[20] K. P. Murphy, Y. Weiss, and M. I. Jordan. Loopy Belief propagation for approximate inference: An empirical study. In *UAI*, pages 467–475, 1999.

[21] M. Niepert. Markov Chains on Orbits of Permutation Groups. In *UAI*, pages 624–633, 2012.

[22] Radford Neal. Slice sampling. *Annals of Statistics*, 31:705–767, 2000.

[23] K. S. Ng, J. W. Lloyd, and W. T. Uther. Probabilistic modelling, inference and learning using logical theories. *Annals of Mathematics and Artificial Intelligence*, 54(1-3):159–205, 2008.

[24] J. Pearl. *Probabilistic Reasoning in Intelligent Systems: Networks of Plausible Inference*. Morgan Kaufmann, San Francisco, CA, 1988.

[25] D. Poole. First-order probabilistic inference. In *IJCAI*, pages 985–991, 2003.

[26] H. Poon and P. Domingos. Sound and efficient inference with probabilistic and deterministic dependencies. In *AAAI*, pages 458–463, 2006.

[27] H. Poon, P. Domingos, and M. Sumner. A general method for reducing the complexity of relational inference and its application to MCMC. In *AAAI*, pages 1075–1080, 2008.

[28] M. Richardson and P. Domingos. Markov logic networks. *Machine Learning*, 62:107–136, 2006.

[29] T. Sang, P. Beame, and H. Kautz. Solving Bayesian networks by weighted model counting. In *AAAI*, pages 475–482, 2005.

[30] P. Singla and P. Domingos. Lifted first-order belief propagation. In *AAAI*, pages 1094–1099, Chicago, IL, 2008. AAAI Press.

